# Co-Training and Expansion: Towards Bridging Theory and Practice

**Maria-Florina Balcan**
Computer Science Dept.
Carnegie Mellon Univ.
Pittsburgh, PA 15213
ninamf@cs.cmu.edu

**Avrim Blum**
Computer Science Dept.
Carnegie Mellon Univ.
Pittsburgh, PA 15213
avrim@cs.cmu.edu

**Ke Yang**
Computer Science Dept.
Carnegie Mellon Univ.
Pittsburgh, PA 15213
yangke@cs.cmu.edu

## Abstract

Co-training is a method for combining labeled and unlabeled data when examples can be thought of as containing two distinct sets of features. It has had a number of practical successes, yet previous theoretical analyses have needed very strong assumptions on the data that are unlikely to be satisfied in practice.

In this paper, we propose a much weaker "expansion" assumption on the underlying data distribution, that we prove is sufficient for iterative co-training to succeed given appropriately strong PAC-learning algorithms on each feature set, and that to some extent is necessary as well. This expansion assumption in fact motivates the iterative nature of the original co-training algorithm, unlike stronger assumptions (such as independence given the label) that allow a simpler one-shot co-training to succeed. We also heuristically analyze the effect on performance of noise in the data. Predicted behavior is qualitatively matched in synthetic experiments on expander graphs.

## 1 Introduction

In machine learning, it is often the case that unlabeled data is substantially cheaper and more plentiful than labeled data, and as a result a number of methods have been developed for using unlabeled data to try to improve performance, e.g., [15, 2, 6, 11, 16]. Co-training [2] is a method that has had substantial success in scenarios in which examples can be thought of as containing two distinct yet sufficient feature sets. Specifically, a labeled example takes the form $(\langle x_1, x_2 \rangle, \ell)$, where $x_1 \in X_1$ and $x_2 \in X_2$ are the two parts of the example, and $\ell$ is the label. One further assumes the existence of two functions $c_1, c_2$ over the respective feature sets such that $c_1(x_1) = c_2(x_2) = \ell$. Intuitively, this means that each example contains two "views," and each view contains sufficient information to determine the label of the example. This redundancy implies an underlying structure of the unlabeled data (since they need to be "consistent"), and this structure makes the unlabeled data informative. In particular, the idea of *iterative co-training* [2] is that one can use a small labeled sample to train initial classifiers $h_1, h_2$ over the respective views, and then iteratively bootstrap by taking unlabeled examples $\langle x_1, x_2 \rangle$ for which one of the $h_i$ is confident but the other is not — and using the confident $h_i$ to label such examples for the learning algorithm on the other view, improving the other classifier. As an example for webpage classification given in [2], webpages contain text ($x_1$) and have hyperlinks pointing to them ($x_2$). From a small labeled sample, we might learn a classifier $h_2$ that says that if a link with the words "my advisor" points to a page, then that page is probably a positive example of faculty-member-home-page; so, if we find an unlabeled example with this property we can use $h_2$ to label the page for the learning algorithm that uses the text on the page itself. This approach and its variants have been used for a variety of learning problems, including

named entity classification [3], text classification [10, 5], natural language processing [13], large scale document classification [12], and visual detectors [8].

Co-training effectively requires two distinct properties of the underlying data distribution in order to work. The first is that there should at least in principle exist low error classifiers $c_1, c_2$ on each view. The second is that these two views should on the other hand not be *too* highly correlated — we need to have at least some examples where $h_1$ is confident but $h_2$ is not (or vice versa) for the co-training algorithm to actually do anything. Unfortunately, previous theoretical analyses have needed to make strong assumptions of this second type in order to prove their guarantees. These include "conditional independence given the label" used by [2] and [4], or the assumption of "weak rule dependence" used by [1]. The primary contribution of this paper is a theoretical analysis that *substantially* relaxes the strength of this second assumption to just a form of "expansion" of the underlying distribution (a natural analog of the graph-theoretic notions of expansion and conductance) that we show in some sense is a necessary condition for co-training to succeed as well. However, we will need a fairly strong assumption on the learning algorithms: that the $h_i$ they produce are never "confident but wrong" (formally, the algorithms are able to learn from positive data only), though we give a heuristic analysis of the case when this does not hold.

One key feature of assuming only expansion on the data is that it specifically motivates the iterative nature of the co-training algorithm. Previous assumptions that had been analyzed imply such a strong form of expansion that even a "one-shot" version of co-training will succeed (see Section 2.2). In fact, the theoretical guarantees given in [2] are exactly of this type. However, distributions can easily satisfy our weaker condition without allowing one-shot learning to work as well, and we describe several natural situations of this form. An additional property of our results is that they are algorithmic in nature. That is, if we have sufficiently strong *efficient* PAC-learning algorithms for the target function on each feature set, we can use them to achieve *efficient* PAC-style guarantees for co-training as well. However, as mentioned above, we need a stronger assumption on our base learning algorithms than used by [2] (see section 2.1).

We begin by formally defining the expansion assumption we will use, connecting it to standard graph-theoretic notions of expansion and conductance. We then prove the statement that $\epsilon$-expansion is sufficient for iterative co-training to succeed, given strong enough base learning algorithms over each view, proving bounds on the number of iterations needed to converge. In Section 4.1, we heuristically analyze the effect of imperfect feature sets on co-training accuracy. Finally, in Section 4.2, we present experiments on synthetic expander graph data that qualitatively bear out our analyses.

## 2 Notations, Definitions, and Assumptions

We assume that examples are drawn from some distribution $D$ over an instance space $X = X_1 \times X_2$, where $X_1$ and $X_2$ correspond to two different "views" of an example. Let $c$ denote the target function, and let $X^+$ and $X^-$ denote the positive and negative regions of $X$ respectively (for simplicity we assume we are doing binary classification). For most of this paper we assume that each view in itself is sufficient for correct classification; that is, $c$ can be decomposed into functions $c_1, c_2$ over each view such that $D$ has no probability mass on examples $x$ such that $c_1(x_1) \neq c_2(x_2)$. For $i \in \{1, 2\}$, let $X_i^+ = \{x_i \in X_i : c_i(x_i) = 1\}$, so we can think of $X^+$ as $X_1^+ \times X_2^+$, and let $X_i^- = X_i - X_i^+$. Let $D^+$ and $D^-$ denote the marginal distribution of $D$ over $X^+$ and $X^-$ respectively.

In order to discuss iterative co-training, we need to be able to talk about a hypothesis being confident or not confident on a given example. For convenience, we will identify "confident" with "confident about being positive". This means we can think of a hypothesis $h_i$ as a subset of $X_i$, where $x_i \in h_i$ means that $h_i$ is confident that $x_i$ is positive, and $x_i \notin h_i$ means that $h_i$ has no opinion.

As in [2], we will abstract away the initialization phase of co-training (how labeled data is used to generate an initial hypothesis) and assume we are given initial sets $S_1^0 \subseteq X_1^+$ and

$S_2^0 \subseteq X_2^+$ such that $\Pr_{\langle x_1, x_2 \rangle \in D}(x_1 \in S_1^0 \text{ or } x_2 \in S_2^0) \geq \rho_{init}$ for some $\rho_{init} > 0$. The goal of co-training will be to bootstrap from these sets using unlabeled data.

Now, to prove guarantees for iterative co-training, we make two assumptions: that the learning algorithms used in each of the two views are able to learn from positive data only, and that the distribution $D^+$ is expanding as defined in Section 2.2 below.

## 2.1 Assumption about the base learning algorithms on the two views

We assume that the learning algorithms on each view are able to PAC-learn from positive data only. Specifically, for any distribution $D_i^+$ over $X_i^+$, and any given $\epsilon, \delta > 0$, given access to examples from $D_i^+$ the algorithm should be able to produce a hypothesis $h_i$ such that (a) $h_i \subseteq X_i^+$ (so $h_i$ only has one-sided error), and (b) with probability $1-\delta$, the error of $h_i$ under $D_i^+$ is at most $\epsilon$. Algorithms of this type can be naturally thought of as predicting either "positive with confidence" or "don't know", fitting our framework. Examples of concept classes learnable from positive data only include conjunctions, k-CNF, and axis-parallel rectangles; see [7]. For instance, for the case of axis-parallel rectangles, a simple algorithm that achieves this guarantee is just to output the smallest rectangle enclosing the positive examples seen.

If we wanted to consider algorithms that could be confident in both directions (rather than just confident about being positive) we could instead use the notion of "reliable, useful" learning due to Rivest and Sloan [14]. However, fewer classes of functions are learnable in this manner. In addition, a nice feature of our assumption is that we will only need $D^+$ to expand and not $D^-$. This is especially natural if the positive class has a large amount of cohesion (e.g, it consists of all documents about some topic $Y$) but the negatives do not (e.g., all documents about all other topics). Note that we are effectively assuming that our algorithms are correct when they are confident; we relax this in our heuristic analysis in Section 4.

## 2.2 The expansion assumption for the underlying distribution

For $S_1 \subseteq X_1$ and $S_2 \subseteq X_2$, let boldface $\mathbf{S}_i$ ($i = 1, 2$) denote the event that an example $\langle x_1, x_2 \rangle$ has $x_i \in S_i$. So, if we think of $S_1$ and $S_2$ as our confident sets in each view, then $\Pr(\mathbf{S}_1 \wedge \mathbf{S}_2)$ denotes the probability mass on examples for which we are confident about both views, and $\Pr(\mathbf{S}_1 \oplus \mathbf{S}_2)$ denotes the probability mass on examples for which we are confident about just one. In this section, all probabilities are with respect to $D^+$. We say:

**Definition 1** $D^+$ is $\epsilon$-**expanding** if for any $S_1 \subseteq X_1^+$, $S_2 \subseteq X_2^+$, we have

$$\Pr(\mathbf{S}_1 \oplus \mathbf{S}_2) \geq \epsilon \min \left[ \Pr(\mathbf{S}_1 \wedge \mathbf{S}_2), \Pr(\overline{\mathbf{S}_1} \wedge \overline{\mathbf{S}_2}) \right].$$

*We say that $D^+$ is $\epsilon$-**expanding with respect to hypothesis class** $H_1 \times H_2$ if the above holds for all $S_1 \in H_1 \cap X_1^+, S_2 \in H_2 \cap X_2^+$ (here we denote by $H_i \cap X_i^+$ the set $\left\{ h \cap X_i^+ : h \in H_i \right\}$ for $i = 1, 2$).*

To get a feel for this definition, notice that $\epsilon$-expansion is in some sense necessary for iterative co-training to succeed, because if $S_1$ and $S_2$ are our confident sets and do not expand, then we might never see examples for which one hypothesis could help the other.[1] In Section 3 we show that Definition 1 is in fact sufficient. To see how much weaker this definition is than previously-considered requirements, it is helpful to consider a slightly stronger kind of expansion that we call "left-right expansion".

**Definition 2** *We say $D^+$ is $\epsilon$-right-expanding if for any $S_1 \subseteq X_1^+, S_2 \subseteq X_2^+$,*

*if $\Pr(\mathbf{S}_1) \leq 1/2$ and $\Pr(\mathbf{S}_2|\mathbf{S}_1) \geq 1 - \epsilon$ then $\Pr(\mathbf{S}_2) \geq (1 + \epsilon) \Pr(\mathbf{S}_1)$.*

*We say $D^+$ is $\epsilon$-left-expanding if the above holds with indices 1 and 2 reversed. Finally, $D^+$ is $\epsilon$-**left-right-expanding** if it has both properties.*

It is not immediately obvious but left-right expansion in fact implies Definition 1 (see Appendix A), though the converse is not necessarily true. We introduce this notion, however, for two reasons. First, it is useful for intuition: if $S_i$ is our confident set in $X_i^+$ and this set is small ($\Pr(\mathbf{S}_i) \leq 1/2$), and we train a classifier that learns from positive data on the conditional distribution that $S_i$ induces over $X_{3-i}$ until it has error $\leq \epsilon$ on that distribution, then the definition implies the confident set on $X_{3-i}$ will have noticeably larger probability than $S_i$; so it is clear why this is useful for co-training, at least in the initial stages. Secondly, this notion helps clarify how our assumptions are much less restrictive than those considered previously. Specifically,

**Independence given the label:** Independence given the label implies that for any $S_1 \subseteq X_1^+$ and $S_2 \subseteq X_2^+$ we have $\Pr(\mathbf{S}_2|\mathbf{S}_1) = \Pr(\mathbf{S}_2)$. So, if $\Pr(\mathbf{S}_2|\mathbf{S}_1) \geq 1 - \epsilon$, then $\Pr(\mathbf{S}_2) \geq 1 - \epsilon$ as well, even if $\Pr(\mathbf{S}_1)$ is tiny. This means that not only does $S_1$ expand by a $(1 + \epsilon)$ factor as in Def. 2, but in fact it *expands to nearly all of $X_2^+$*.

**Weak dependence:** Weak dependence [1] is a relaxation of conditional independence that requires only that for all $S_1 \subseteq X_1^+, S_2 \subseteq X_2^+$ we have $\Pr(\mathbf{S}_2|\mathbf{S}_1) \geq \alpha \Pr(\mathbf{S}_2)$ for some $\alpha > 0$. This seems much less restrictive. However, notice that if $\Pr(\mathbf{S}_2|\mathbf{S}_1) \geq 1 - \epsilon$, then $\Pr(\overline{\mathbf{S}_2}|\mathbf{S}_1) \leq \epsilon$, which implies by definition of weak dependence that $\Pr(\overline{\mathbf{S}_2}) \leq \epsilon/\alpha$ and therefore $\Pr(\mathbf{S}_2) \geq 1 - \epsilon/\alpha$. So, again (for sufficiently small $\epsilon$), even if $S_1$ is very small, it expands to nearly all of $X_2^+$. This means that, as with conditional independence, if one has an algorithm over $X_2$ that PAC-learns from positive data only, and one trains it over the conditional distribution given by $\mathbf{S}_1$, then by driving down its error on this conditional distribution one can perform co-training in just one iteration.

### 2.2.1 Connections to standard graph-theoretic notions of expansion

Our definition of $\epsilon$-expansion (Definition 1) is a natural analog of the standard graph-theoretic notion of *edge-expansion* or *conductance*. A Markov-chain is said to have high conductance if under the stationary distribution, for any set of states $S$ of probability at most $1/2$, the probability mass on transitions exiting $S$ is at least $\epsilon$ times the probability of $S$. E.g., see [9]. A graph has high edge-expansion if the random walk on the graph has high conductance. Since the stationary distribution of this walk can be viewed as having equal probability on every edge, this is equivalent to saying that for any partition of the graph into two pieces $(S, V - S)$, the number of edges crossing the partition should be at least an $\epsilon$ fraction of the number of edges in the smaller half. To connect this to Definition 1, think of $S$ as $\mathbf{S}_1 \wedge \mathbf{S}_2$.

It is well-known that, for example, a random degree-3 bipartite graph with high probability is expanding, and this in fact motivates our synthetic data experiments of Section 4.2.

### 2.2.2 Examples

We now give two simple examples that satisfy $\epsilon$-expansion but not weak dependence.

**Example 1:** Suppose $X = R^d \times R^d$ and the target function on each view is an axis-parallel rectangle. Suppose a random positive example from $D^+$ looks like a pair $\langle x_1, x_2 \rangle$ such that $x_1$ and $x_2$ are each uniformly distributed in their rectangles but in a *highly-dependent* way: specifically, $x_2$ is *identical* to $x_1$ except that a random coordinate has been "re-randomized" within the rectangle. This distribution does not satisfy weak dependence (for any sets $S$ and $T$ that are disjoint along all axes we have $\Pr(\mathbf{T}|\mathbf{S}) = 0$) but it is not hard to verify that $D^+$ is $\epsilon$-expanding for $\epsilon = \Omega(1/d)$.

**Example 2:** Imagine that we have a learning problem such that the data in $X_1$ falls into $n$ different clusters: the positive class is the union of some of these clusters and the negative class is the union of the others. Imagine that this likewise is true if we look at $X_2$ and for simplicity suppose that every cluster has the same probability mass. Independence given

the label would say that given that $x_1$ is in some positive cluster $C_i$ in $X_1$, $x_2$ is equally likely to be in any of the positive clusters $C_j$ in $X_2$. But, suppose we have something much weaker: each $C_i$ in $X_1$ is associated with only 3 $C_j$'s in $X_2$ (i.e., given that $x_1$ is in $C_i$, $x_2$ will only be in one of these $C_j$'s). This distribution clearly will not even have the weak dependence property. However, say we have a learning algorithm that assumes everything in the same cluster has the same label (so the hypothesis space $H$ consists of all rules that do not split clusters). Then if the graph of which clusters are associated with which is an expander graph, then the distributions will be expanding with respect to $H$. In particular, given a labeled example $x$, the learning algorithm will generalize to $x$'s entire cluster $C_i$, then this will be propagated over to nodes in the associated clusters $C_j$ in $X_2$, and so on.

## 3 The Main Result

We now present our main result. We assume that $D^+$ is $\epsilon$-**expanding** ($\epsilon > 0$) with respect to hypothesis class $H_1 \times H_2$, that we are given initial confident sets $S_1^0 \subseteq X_1^+$, $S_2^0 \subseteq X_2^+$ such that $\Pr(\mathbf{S}_1^0 \vee \mathbf{S}_2^0) \geq \rho_{init}$, that the target function can be written as $\langle c_1, c_2 \rangle$ with $c_1 \in H_1$, $c_2 \in H_2$, and that on each of the two views we have algorithms $\mathcal{A}_1$ and $\mathcal{A}_2$ for learning from positive data only.

The iterative co-training that we consider proceeds in *rounds*. Let $S_1^i \subseteq X_1$ and $S_2^i \subseteq X_2$ be the confident sets in each view at the start of round $i$. We construct $S_2^{i+1}$ by feeding into $\mathcal{A}_2$ examples according to $D_2$ conditioned on $\mathbf{S}_1^i \vee \mathbf{S}_2^i$. That is, we take unlabeled examples from $D$ such that at least one of the current predictors is confident, and feed them into $\mathcal{A}_2$ as if they were positive. We run $\mathcal{A}_2$ with error and confidence parameters given in the theorem below. We simultaneously do the same with $\mathcal{A}_1$, creating $S_1^{i+1}$.

After a pre-determined number of rounds $N$ (specified in Theorem 1), the algorithm terminates and outputs the predictor that labels examples $\langle x_1, x_2 \rangle$ as positive if $x_1 \in S_1^{N+1}$ or $x_2 \in S_2^{N+1}$ and negative otherwise.

We begin by stating two lemmas that will be useful in our analysis. For both of these lemmas, let $S_1, T_1 \subseteq X_1^+$, $S_2, T_2 \subseteq X_2^+$, where $S_j, T_j \in H_j$. All probabilities are with respect to $D^+$.

**Lemma 1** *Suppose* $\Pr(\mathbf{S}_1 \wedge \mathbf{S}_2) \leq \Pr(\overline{\mathbf{S}_1} \wedge \overline{\mathbf{S}_2})$, $\Pr(\mathbf{T}_1 \mid \mathbf{S}_1 \vee \mathbf{S}_2) \geq 1 - \epsilon/8$ *and* $\Pr(\mathbf{T}_2 \mid \mathbf{S}_1 \vee \mathbf{S}_2) \geq 1 - \epsilon/8$. *Then* $\Pr(\mathbf{T}_1 \wedge \mathbf{T}_2) \geq (1 + \epsilon/2) \Pr(\mathbf{S}_1 \wedge \mathbf{S}_2)$.

**Proof:** From $\Pr(\mathbf{T}_1 \mid \mathbf{S}_1 \vee \mathbf{S}_2) \geq 1 - \epsilon/8$ and $\Pr(\mathbf{T}_2 \mid \mathbf{S}_1 \vee \mathbf{S}_2) \geq 1 - \epsilon/8$ we get that $\Pr(\mathbf{T}_1 \wedge \mathbf{T}_2) \geq (1 - \epsilon/4) \Pr(\mathbf{S}_1 \vee \mathbf{S}_2)$. Since $\Pr(\mathbf{S}_1 \wedge \mathbf{S}_2) \leq \Pr(\overline{\mathbf{S}_1} \wedge \overline{\mathbf{S}_2})$ it follows from the expansion property that

$$\Pr(\mathbf{S}_1 \vee \mathbf{S}_2) = \Pr(\mathbf{S}_1 \oplus \mathbf{S}_2) + \Pr(\mathbf{S}_1 \wedge \mathbf{S}_2) \geq (1 + \epsilon) \Pr(\mathbf{S}_1 \wedge \mathbf{S}_2).$$

Therefore, $\Pr(\mathbf{T}_1 \wedge \mathbf{T}_2) \geq (1 - \epsilon/4)(1 + \epsilon) \Pr(\mathbf{S}_1 \wedge \mathbf{S}_2)$ which implies that $\Pr(\mathbf{T}_1 \wedge \mathbf{T}_2) \geq (1 + \epsilon/2) \Pr(\mathbf{S}_1 \wedge \mathbf{S}_2)$. ∎

**Lemma 2** *Suppose* $\Pr(\mathbf{S}_1 \wedge \mathbf{S}_2) > \Pr(\overline{\mathbf{S}_1} \wedge \overline{\mathbf{S}_2})$ *and let* $\gamma = 1 - \Pr(\mathbf{S}_1 \wedge \mathbf{S}_2)$. *If* $\Pr(\mathbf{T}_1 \mid \mathbf{S}_1 \vee \mathbf{S}_2) \geq 1 - \frac{\gamma\epsilon}{8}$ *and* $\Pr(\mathbf{T}_2 \mid \mathbf{S}_1 \vee \mathbf{S}_2) \geq 1 - \frac{\gamma\epsilon}{8}$, *then* $\Pr(\mathbf{T}_1 \wedge \mathbf{T}_2) \geq (1 + \frac{\gamma\epsilon}{8}) \Pr(\mathbf{S}_1 \wedge \mathbf{S}_2)$.

**Proof:** From $\Pr(\mathbf{T}_1 \mid \mathbf{S}_1 \vee \mathbf{S}_2) \geq 1 - \frac{\gamma\epsilon}{8}$ and $\Pr(\mathbf{T}_2 \mid \mathbf{S}_1 \vee \mathbf{S}_2) \geq 1 - \frac{\gamma\epsilon}{8}$ we get that $\Pr(\mathbf{T}_1 \wedge \mathbf{T}_2) \geq (1 - \frac{\gamma\epsilon}{4}) \Pr(\mathbf{S}_1 \vee \mathbf{S}_2)$. Since $\Pr(\mathbf{S}_1 \wedge \mathbf{S}_2) > \Pr(\overline{\mathbf{S}_1} \wedge \overline{\mathbf{S}_2})$ it follows from the expansion property that $\Pr(\mathbf{S}_1 \oplus \mathbf{S}_2) \geq \epsilon \Pr(\overline{\mathbf{S}_1} \wedge \overline{\mathbf{S}_2})$. Therefore

$$\gamma = \Pr(\mathbf{S}_1 \oplus \mathbf{S}_2) + \Pr(\overline{\mathbf{S}_1} \wedge \overline{\mathbf{S}_2}) \geq (1 + \epsilon) \Pr(\overline{\mathbf{S}_1} \wedge \overline{\mathbf{S}_2}) \geq (1 + \epsilon)(1 - \Pr(\mathbf{S}_1 \vee \mathbf{S}_2))$$

and so $\Pr(\mathbf{S}_1 \vee \mathbf{S}_2) \geq 1 - \frac{\gamma}{1+\epsilon}$. This implies $\Pr(\mathbf{T}_1 \wedge \mathbf{T}_2) \geq (1 - \frac{\gamma\epsilon}{4})(1 - \frac{\gamma}{1+\epsilon}) \geq (1 - \gamma)(1 + \frac{\gamma\epsilon}{8})$. So, we have $\Pr(\mathbf{T}_1 \wedge \mathbf{T}_2) \geq (1 + \frac{\gamma\epsilon}{8}) \Pr(\mathbf{S}_1 \wedge \mathbf{S}_2)$. ∎

**Theorem 1** *Let $\epsilon_{fin}$ and $\delta_{fin}$ be the (final) desired accuracy and confidence parameters. Then we can achieve error rate $\epsilon_{fin}$ with probability $1 - \delta_{fin}$ by running co-training for $N = O(\frac{1}{\epsilon} \log \frac{1}{\epsilon_{fin}} + \frac{1}{\epsilon} \cdot \frac{1}{\rho_{init}})$ rounds, each time running $\mathcal{A}_1$ and $\mathcal{A}_2$ with accuracy and confidence parameters set to $\frac{\epsilon \cdot \epsilon_{fin}}{8}$ and $\frac{\delta_{fin}}{2N}$ respectively.*

**Proof Sketch:** Assume that, for $i \geq 1$, $S_1^i \subseteq X_1^+$ and $S_2^i \subseteq X_2^+$ are the confident sets in each view after step $i - 1$ of co-training. Define $p_i = \Pr\left(\mathbf{S}_1^i \wedge \mathbf{S}_2^i\right)$, $q_i = \Pr\left(\overline{\mathbf{S}_1^i} \wedge \overline{\mathbf{S}_2^i}\right)$, and $\gamma_i = 1 - p_i$, with all probabilities with respect to $D^+$. We are interested in bounding $\Pr\left(\mathbf{S}_1^i \vee \mathbf{S}_2^i\right)$, but since technically it is easier to bound $\Pr\left(\mathbf{S}_1^i \wedge \mathbf{S}_2^i\right)$, we will instead show that $p_N \geq 1 - \epsilon_{fin}$ with probability $1 - \delta_{fin}$, which obviously implies that $\Pr(\mathbf{S}_1^N \vee \mathbf{S}_2^N)$ is at least as good.

By the guarantees on $\mathcal{A}_1$ and $\mathcal{A}_2$, after each round we get that with probability $1 - \frac{\delta_{fin}}{N}$, we have $\Pr\left(\mathbf{S}_1^{i+1} \mid \mathbf{S}_1^i \vee \mathbf{S}_2^i\right) \geq 1 - \frac{\epsilon_{fin} \cdot \epsilon}{8}$ and $\Pr\left(\mathbf{S}_2^{i+1} \mid \mathbf{S}_1^i \vee \mathbf{S}_2^i\right) \geq 1 - \frac{\epsilon_{fin} \cdot \epsilon}{8}$. In particular, this implies that with probability $1 - \frac{\delta_{fin}}{N}$, we have $p_1 = \Pr\left(\mathbf{S}_1^1 \wedge \mathbf{S}_2^1\right) \geq (1 - \epsilon/4) \cdot \Pr\left(\mathbf{S}_1^0 \vee \mathbf{S}_2^0\right) \geq (1 - \epsilon/4)\rho_{init}$.

Consider now $i \geq 1$. If $p_i \leq q_i$, since with probability $1 - \frac{\delta_{fin}}{N}$ we have $\Pr\left(\mathbf{S}_1^{i+1} \mid \mathbf{S}_1^i \vee \mathbf{S}_2^i\right) \geq 1 - \frac{\epsilon}{8}$ and $\Pr\left(\mathbf{S}_2^{i+1} \mid \mathbf{S}_1^i \vee \mathbf{S}_2^i\right) \geq 1 - \frac{\epsilon}{8}$, using lemma 1 we obtain that with probability $1 - \frac{\delta_{fin}}{N}$, we have $\Pr\left(\mathbf{S}_1^{i+1} \wedge \mathbf{S}_2^{i+1}\right) \geq (1 + \epsilon/2)\Pr\left(\mathbf{S}_1^i \wedge \mathbf{S}_2^i\right)$. Similarly, by applying lemma 2, we obtain that if $p_i > q_i$ and $\gamma_i \geq \epsilon_{fin}$ then with probability $1 - \frac{\delta_{fin}}{N}$ we have $\Pr\left(\mathbf{S}_1^{i+1} \wedge \mathbf{S}_2^{i+1}\right) \geq (1 + \frac{\gamma_i \epsilon}{8})\Pr\left(\mathbf{S}_1^i \wedge \mathbf{S}_2^i\right)$. Assume now that it is the case that the learning algorithms $\mathcal{A}_1$ and $\mathcal{A}_2$ were successful on all the $N$ rounds; note that this happens with probability at least $1 - \delta_{fin}$.

The above observations imply that so long as $p_i \leq 1/2$ (so $\gamma_i \geq 1/2$) we have $p_{i+1} \geq (1 + \epsilon/16)^i (1 - \epsilon/4)\rho_{init}$. This means that after $N_1 = O(\frac{1}{\rho_{init}} \cdot \frac{1}{\epsilon})$ iterations of co-training we get to a situation where $p_{N_1} > 1/2$. At this point, notice that every $8/\epsilon$ rounds, $\gamma$ drops by at least a factor of 2; that is, if $\gamma_i \leq \frac{1}{2^k}$ then $\gamma_{\frac{8}{\epsilon}+i} \leq \frac{1}{2^{k+1}}$. So, after a total of $O(\frac{1}{\epsilon} \log \frac{1}{\epsilon_{fin}} + \frac{1}{\epsilon} \cdot \frac{1}{\rho_{init}})$ rounds, we have a predictor of the desired accuracy with the desired confidence. ∎

## 4 Heuristic Analysis of Error propagation and Experiments

So far, we have assumed the existence of perfect classifiers on each view: there are no examples $\langle x_1, x_2 \rangle$ with $x_1 \in X_1^+$ and $x_2 \in X_2^-$ or vice-versa. In addition, we have assumed that given correctly-labeled positive examples as input, our learning algorithms are able to generalize in a way that makes only 1-sided error (i.e., they are never "confident but wrong"). In this section we give a heuristic analysis of the case when these assumptions are relaxed, along with several synthetic experiments on expander graphs.

### 4.1 Heuristic Analysis of Error propagation

Given confident sets $S_1^i \subseteq X_1$ and $S_2^i \subseteq X_2$ at the $i$th iteration, let us define their *purity* (precision) as $pur_i = \Pr_D(c(x) = 1 | \mathbf{S}_1^i \vee \mathbf{S}_2^i)$ and their *coverage* (recall) to be $cov_i = \Pr_D(\mathbf{S}_1^i \vee \mathbf{S}_2^i | c(x) = 1)$. Let us also define their "opposite coverage" to be $opp_i = \Pr_D(\mathbf{S}_1^i \vee \mathbf{S}_2^i | c(x) = 0)$. Previously, we assumed $opp_i = 0$ and therefore $pur_i = 1$. However, if we imagine that there is an $\eta$ fraction of examples on which the two views disagree, and that positive and negative regions expand uniformly at the same rate, then even if initially $opp_0 = 0$, it is natural to assume the following form of increase in $cov$ and $opp$:

$$cov_{i+1} = \min\left(cov_i(1 + \epsilon(1 - cov_i)) + \eta \cdot (opp_{i+1} - opp_i), 1\right), \qquad (1)$$
$$opp_{i+1} = \min\left(opp_i(1 + \epsilon(1 - opp_i)) + \eta \cdot (cov_{i+1} - cov_i), 1\right). \qquad (2)$$

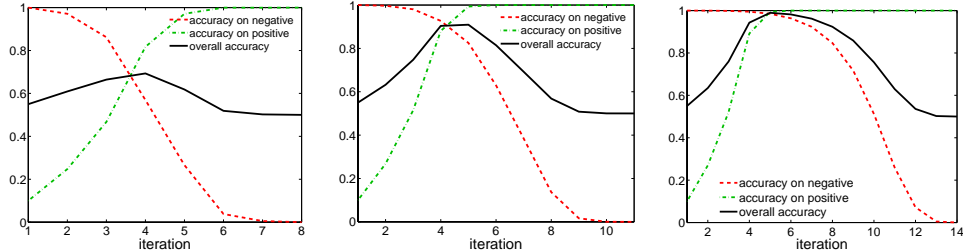

Figure 1: Co-training with noise rates 0.1, 0.01, and 0.001 respectively ($n = 5000$). Solid line indicates overall accuracy; green (dashed, increasing) curve is accuracy on positives ($cov_i$); red (dashed, decreasing) curve is accuracy on negatives ($1 - opp_i$).

That is, this corresponds to both the positive and negative parts of the confident region expanding in the way given in the proof of Theorem 1, with an $\eta$ fraction of the new edges going to examples of the other label. By examining (1) and (2), we can make a few simple observations. First, initially when coverage is low, every $O(1/\epsilon)$ steps we get roughly $cov \leftarrow 2 \cdot cov$ and $opp \leftarrow 2 \cdot opp + \eta \cdot cov$. So, we expect coverage to increase exponentially and purity to drop linearly. However, once coverage gets large and begins to saturate, if purity is still high at this time it will begin dropping rapidly as the exponential increase in $opp_i$ causes $opp_i$ to catch up with $cov_i$. In particular, a calculation (omitted) shows that if $D$ is 50/50 positive and negative, then overall accuracy increases up to the point when $cov_i + opp_i = 1$, and then drops from then on. This qualitative behavior is borne out in our experiments below.

## 4.2 Experiments

We performed experiments on synthetic data along the lines of Example 2, with noise added as in Section 4.1. Specifically, we create a $2n$-by-$2n$ bipartite graph. Nodes 1 to $n$ on each side represent positive clusters, and nodes $n + 1$ to $2n$ on each side represent negative clusters. We connect each node on the left to three nodes on the right: each neighbor is chosen with probability $1 - \eta$ to be a random node of the same class, and with probability $\eta$ to be a random node of the opposite class. We begin with an initial confident set $S_1 \subseteq X_1^+$ and then propagate confidence through rounds of co-training, monitoring the percentage of the positive class covered, the percent of the negative class mistakenly covered, and the overall accuracy. Plots of three experiments are shown in Figure 1, for different noise rates (0.1, 0.01, and 0.001). As can be seen, these qualitatively match what we expect: coverage increases exponentially, but accuracy on negatives ($1 - opp_i$) drops exponentially too, though somewhat delayed. At some point there is a crossover where $cov_i = 1 - opp_i$, which as predicted roughly corresponds to the point at which overall accuracy starts to drop.

## 5 Conclusions

Co-training is a method for using unlabeled data when examples can be partitioned into two views such that (a) each view in itself is at least roughly sufficient to achieve good classification, and yet (b) the views are not too highly correlated. Previous theoretical work has required instantiating condition (b) in a very strong sense: as independence given the label, or a form of weak dependence. In this work, we argue that the "right" condition is something much weaker: an expansion property on the underlying distribution (over positive examples) that we show is sufficient and to some extent necessary as well.

The expansion property is especially interesting because it directly motivates the iterative nature of many of the practical co-training based algorithms, and our work is the first rigorous analysis of iterative co-training in a setting that demonstrates its advantages over one-shot versions.

**Acknowledgements:** This work was supported in part by NSF grants CCR-0105488, NSF-ITR CCR-0122581, and NSF-ITR IIS-0312814.

## Footnotes

[1] However, $\epsilon$-expansion requires *every* pair to expand and so it is not strictly necessary. If there were occasional pairs $(S_1, S_2)$ that did not expand, but such pairs were rare and unlikely to be encountered as confident sets in the co-training process, we might still be OK.

# References

[1] S. Abney. Bootstrapping. In *Proceedings of the 40th Annual Meeting of the Association for Computational Linguistics (ACL)*, pages 360–367, 2002.

[2] A. Blum and T. M. Mitchell. Combining labeled and unlabeled data with co-training. In *Proc. 11th Annual Conference on Computational Learning Theory*, pages 92–100, 1998.

[3] M. Collins and Y. Singer. Unsupervised models for named entity classification. In *SIGDAT Conf. Empirical Methods in NLP and Very Large Corpora*, pages 189–196, 1999.

[4] S. Dasgupta, M. L. Littman, and D. McAllester. PAC generalization bounds for co-training. In *Advances in Neural Information Processing Systems 14*. MIT Press, 2001.

[5] R. Ghani. Combining labeled and unlabeled data for text classification with a large number of categories. In *Proceedings of the IEEE International Conference on Data Mining*, 2001.

[6] T. Joachims. Transductive inference for text classification using support vector machines. In *Proceedings of the 16th International Conference on Machine Learning*, pages 200–209, 1999.

[7] M. Kearns, M. Li, and L. Valiant. Learning Boolean formulae. *JACM*, 41(6):1298–1328, 1995.

[8] A. Levin, Paul Viola, and Yoav Freund. Unsupervised improvement of visual detectors using co-training. In *Proc. 9th IEEE International Conf. on Computer Vision*, pages 626–633, 2003.

[9] R. Motwani and P. Raghavan. *Randomized Algorithms*. Cambridge University Press, 1995.

[10] K. Nigam and R. Ghani. Analyzing the effectiveness and applicability of co-training. In *Proc. ACM CIKM Int. Conf. on Information and Knowledge Management*, pages 86–93, 2000.

[11] K. Nigam, A. McCallum, S. Thrun, and T. M. Mitchell. Text classification from labeled and unlabeled documents using em. *Machine Learning*, 39(2/3):103–134, 2000.

[12] S. Park and B. Zhang. Large scale unstructured document classification using unlabeled data and syntactic information. In *PAKDD 2003*, LNCS vol. 2637, pages 88–99. Springer, 2003.

[13] D. Pierce and C. Cardie. Limitations of Co-Training for natural language learning from large datasets. In *Proc. Conference on Empirical Methods in NLP*, pages 1–9, 2001.

[14] R. Rivest and R. Sloan. Learning complicated concepts reliably and usefully. In *Proceedings of the 1988 Workshop on Computational Learning Theory*, pages 69–79, 1988.

[15] David Yarowsky. Unsupervised word sense disambiguation rivaling supervised methods. In *Meeting of the Association for Computational Linguistics*, pages 189–196, 1995.

[16] X. Zhu, Z. Ghahramani, and J. Lafferty. Semi-supervised learning using gaussian fields and harmonic functions. In *Proc. 20th International Conf. Machine Learning*, pages 912–912, 2003.

## A  Relating the definitions

We show here how Definition 2 implies Definition 1.

**Theorem 2** *If $D^+$ satisfies $\epsilon$-left-right expansion (Definition 2), then it also satisfies $\epsilon'$-expansion (Definition 1) for $\epsilon' = \epsilon/(1+\epsilon)$.*

**Proof:** We will prove the contrapositive. Suppose there exist $S_1 \subseteq X_1^+, S_2 \subseteq X_2^+$ such that $\Pr(\mathbf{S}_1 \oplus \mathbf{S}_2) < \epsilon' \min\left[\Pr(\mathbf{S}_1 \wedge \mathbf{S}_2), \Pr(\overline{\mathbf{S}_1} \wedge \overline{\mathbf{S}_2})\right]$. Assume without loss of generality that $\Pr(\mathbf{S}_1 \wedge \mathbf{S}_2) \leq \Pr(\overline{\mathbf{S}_1} \wedge \overline{\mathbf{S}_2})$. Since $\Pr(\mathbf{S}_1 \wedge \mathbf{S}_2) + \Pr(\overline{\mathbf{S}_1} \wedge \overline{\mathbf{S}_2}) + \Pr(\mathbf{S}_1 \oplus \mathbf{S}_2) = 1$ it follows that $\Pr(\mathbf{S}_1 \wedge \mathbf{S}_2) \leq \frac{1}{2} - \frac{\Pr(\mathbf{S}_1 \oplus \mathbf{S}_2)}{2}$. Assume $\Pr(\mathbf{S}_1) \leq \Pr(\mathbf{S}_2)$. This implies that $\Pr(\mathbf{S}_1) \leq \frac{1}{2}$ since $\Pr(\mathbf{S}_1) + \Pr(\mathbf{S}_2) = 2\Pr(\mathbf{S}_1 \wedge \mathbf{S}_2) + \Pr(\mathbf{S}_1 \oplus \mathbf{S}_2)$ and so $\Pr(\mathbf{S}_1) \leq \Pr(\mathbf{S}_1 \wedge \mathbf{S}_2) + \frac{\Pr(\mathbf{S}_1 \oplus \mathbf{S}_2)}{2}$. Now notice that

$$\Pr(\mathbf{S}_2|\mathbf{S}_1) = \frac{\Pr(\mathbf{S}_1 \wedge \mathbf{S}_2)}{\Pr(\mathbf{S}_1)} \geq \frac{\Pr(\mathbf{S}_1 \wedge \mathbf{S}_2)}{\Pr(\mathbf{S}_1 \wedge \mathbf{S}_2) + \Pr(\mathbf{S}_1 \oplus \mathbf{S}_2)} > \frac{1}{1+\epsilon'} \geq 1 - \epsilon.$$

But

$$\Pr(\mathbf{S}_2) \leq \Pr(\mathbf{S}_1 \wedge \mathbf{S}_2) + \Pr(\mathbf{S}_1 \oplus \mathbf{S}_2) < (1+\epsilon')\Pr(\mathbf{S}_1 \wedge \mathbf{S}_2) \leq (1+\epsilon)\Pr(\mathbf{S}_1)$$

and so $\Pr(\mathbf{S}_2) < (1+\epsilon)\Pr(\mathbf{S}_1)$. Similarly if $\Pr(\mathbf{S}_2) \leq \Pr(\mathbf{S}_1)$ we get a failure of expansion in the other direction. This completes the proof. ∎